# Recurrent cortical competition: Strengthen or weaken?

**Péter Adorján\*, Lars Schwabe,**
**Christian Piepenbrock\*, and Klaus Obermayer**

Dept. of Comp. Sci., FR2-1, Technical University Berlin
Franklinstrasse 28/29 10587 Berlin, Germany
adorjan@epigenomics.com, {schwabe, oby}@cs.tu-berlin.de,
piepenbrock@epigenomics.com
http://www.ni.cs.tu-berlin.de

## Abstract

We investigate the short term dynamics of the recurrent competition and neural activity in the primary visual cortex in terms of information processing and in the context of orientation selectivity. We propose that after stimulus onset, the strength of the recurrent excitation decreases due to fast synaptic depression. As a consequence, the network shifts from an initially highly nonlinear to a more linear operating regime. Sharp orientation tuning is established in the first highly competitive phase. In the second and less competitive phase, precise signaling of multiple orientations and long range modulation, e.g., by intra- and inter-areal connections becomes possible (surround effects). Thus the network first extracts the salient features from the stimulus, and then starts to process the details. We show that this signal processing strategy is optimal if the neurons have limited bandwidth and their objective is to transmit the maximum amount of information in any time interval beginning with the stimulus onset.

## 1  Introduction

In the last four decades there has been a vivid and highly polarized discussion about the role of recurrent competition in the primary visual cortex (V1) (see [12] for review). The main question is whether the recurrent excitation sharpens a weakly orientation tuned feed-forward input, or the feed-forward input is already sharply tuned, hence the massive recurrent circuitry has a different function. Strong cortical recurrency implements a highly nonlinear mapping of the feed-forward input, and obtains robust and sharply tuned cortical response even if only a weak or no feed-forward orientation bias is present [6, 11, 2]. However, such a competitive network in most cases fails to process multiple orientations within the classical receptive field and may signal spurious orientations [7]. This motivates the concept that the primary visual cortex maps an already sharply orientation tuned feed-forward input in a less competitive (more linear) fashion [9, 13].

Although these models for orientation selectivity in V1 vary on a wide scale, they have one common feature: each of them assumes that the synaptic strength is constant on the short time scale on which the network operates. Given the phenomenon of fast synaptic

dynamics this, however, does not need to be the case. Short term synaptic dynamics, e.g., of the recurrent excitatory synapses would allow a cortical network to operate in both—competitive and linear—regimes. We will show below (Section 2) that such a *dynamic cortical amplifier* network can establish sharp contrast invariant orientation tuning from a broadly tuned feed-forward input, while it is still able to respond correctly to multiple orientations.

We then show (Section 3) that decreasing the recurrent competition with time naturally follows from functional considerations, i.e. from the requirement that the mutual information between stimuli and representations is maximal for any time interval beginning with stimulus onset. We consider a free-viewing scenario, where the cortical layer represents a series of static images that are flashed onto the retina for a fixation period ($\Delta T = 200 - 300$ ms) between saccades. We also assume that the spike count in increasing time windows after stimulus onset carries the information. The key observations are that the signal-to-noise ratio of the cortical representation increases with time (because more spikes are available) and that the optimal strength of the recurrent connections (w.r.t. information transfer) decreases with the decreasing output noise. Consequently the model predicts that the information content per spike (or the SNR for a *fixed sliding* time window) decreases with time for a flashed static stimulus in accordance with recent experimental studies. The neural system thus adapts to its own internal changes by modifying its coding strategy, a phenomenon which one may refer to as *"dynamic coding"*.

## 2    Cortical amplifier with fast synaptic plasticity

To investigate our first hypothesis, we set up a model for an orientation-hypercolumn in the primary visual cortex with similar structure and parameters as in [7]. The important novel feature of our model is that fast synaptic depression is present at the recurrent excitatory connections. Neurons in the cortical layer receive orientation-tuned feed-forward input from the LGN and they are connected via a Mexican-hat shaped recurrent kernel in orientation space. In addition, the recurrent and feed-forward excitatory synapses exhibit fast depression due to the activity dependent depletion of the synaptic transmitter [1, 14]. We compare the response of the cortical amplifier models with and without fast synaptic plasticity at the recurrent excitatory connections to single and multiple bars within the classical receptive field.

The membrane potential $V(\theta, t)$ of a cortical cell tuned to an orientation $\theta$ decreases due to the leakage and the recurrent inhibition, and increases due to the recurrent excitation

$$\tau \frac{\partial}{\partial t} V(\theta, t) + V(\theta, t) = I^{\text{LGN}}(\theta, t) + I^{\text{exc}}(\theta, t) - I^{\text{inh}}(\theta, t), \tag{1}$$

where $\tau = 15$ ms is the membrane time constant and $I^{\text{LGN}}(\theta, t)$ is the input received from the LGN. The recurrent excitatory and inhibitory cortical inputs are given by

$$I^{\alpha}(\theta, t) = \int_{-\frac{\pi}{2}}^{+\frac{\pi}{2}} J^{\alpha}(\theta, \theta', t) \exp\left(-\frac{\Delta(\theta', \theta)^2}{2\sigma_{\alpha}^2}\right) f(\theta', t) \, d\theta' \tag{2}$$

where $\Delta(\theta', \theta)$ is a $\pi$ periodic circular difference between the preferred orientations, $J^{\alpha}(\theta, \theta', t)$ are the excitatory and inhibitory connection strengths (with $\alpha \in \{\text{exc}, \text{inh}\}$, $J_{\max}^{\text{exc}} = 0.2$ mV/Hz and $J_{\max}^{\text{inh}} = 0.8$mV/Hz), and $f$ is the presynaptic firing rate. The excitatory synaptic efficacy $J^{\text{exc}}$ is time dependent due to the fast synaptic depression, while the efficacy of inhibitory synapses $J^{\text{inh}}$ is assumed to be constant. The recurrent excitation is sharply tuned $\sigma_{\text{exc}} = 7.5°$, while the inhibition has broad tuning $\sigma_{\text{inh}} = 90°$. The mapping from the membrane potential to firing rate is approximated by a linear function with a threshold at 0 ($f(\theta) = \beta \max(0, V(\theta))$, $\beta = 15$Hz/mV). Gaussian-noise with variances

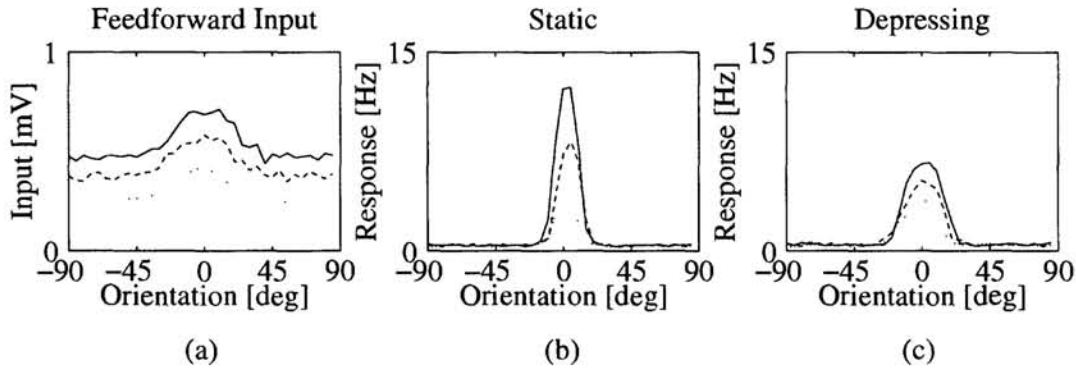

Figure 1: The feed-forward input (a), and the response of the cortical amplifier model with static recurrent synaptic strength (b), and a network with fast synaptic depression (c) if the stimulus is single bar with different stimulus contrasts (40%dotted; 60%dashed; 80%solid line). The cortical response is averaged over the first 100 ms after stimulus onset.

of 6 Hz and 1.6 Hz is added to the input intensities and to the output of cortical neurons. The orientation tuning curves of the feed-forward input $I^{LGN}$ are Gaussians ($\sigma_{LGN} = 18°$) resting on a strong additive orientation independent component which would correspond to a geniculo-cortical connectivity pattern with an approximate aspect ratio of 1:2. Both, the orientation dependent and independent components increase with contrast. Considering a free-viewing scenario where the environment is scanned by saccading around and fixating for short periods of $200 - 300$ ms we model stationary stimuli present for 300 ms. The stimuli are one or more bars with different orientations.

Feed-forward and recurrent excitatory synapses exhibit fast depression. Fast synaptic depression is modeled by the dynamics of the expected synaptic transmitter or "resource" $\bar{R}(t)$ for each synapse. The amount of the available transmitter decreases proportionally to the release probability $p$ and to the presynaptic firing rate $f$, and it recovers exponentially ($\tau_{rec}^{LGN} = 120$ ms, $\tau_{rec}^{Ctx} = 850$ ms, $p^{LGN} = 0.35$ and $p^{Ctx} = 0.55$),

$$\frac{d}{dt}\bar{R}(t) = \frac{1 - \bar{R}(t)}{\tau_{rec}} - f(t)p(t)\bar{R}(t) = -\frac{\bar{R}(t)}{\tau_{eff}(f(t), p(t))} + \frac{1}{\tau_{rec}}. \qquad (3)$$

The change of the membrane potential on the postsynaptic cell at time $t$ is proportional to the released transmitter $pR(t)$. The excitatory connectivity strength between neurons tuned to orientations $\theta$ and $\theta'$ is expressed as $J^{exc}(\theta, \theta', t) = J_{max}^{exc}pR_{\theta\theta'}(t)$. Similarly this applies to the feed-forward synapses. Fast synaptic plasticity at the feed-forward synapses has been investigated in more detail in previous studies [3, 4].

In the following, we compare the predictions of the cortical amplifier model with and without fast synaptic depression at the recurrent excitatory connections. In both cases fast synaptic depression is present at the feed-forward connections limiting the duration of the effective feed-forward input to $200 - 400$ ms. Figure 1 shows the orientation tuning curves at different stimulus contrasts. The feed-forward input is noisy and broadly tuned (Fig. 1a). Both models exhibit contrast invariant tuning (Fig. 1b, c). If fast synaptic depression is present at the recurrent excitation, the cortical network sharpens the broadly tuned feed-forward input in the initial response phase. Once sharply tuned input is established, the tuning width does not change, only the response amplitude decreases in time.

The predictions of the two models differ substantially if multiple orientations are present (Fig. 2). At first, we test the cortical response to two bars separated by 60° with different intensities (Figs. 2a, b). If the recurrent synaptic weights are static and strong enough (Fig. 2a), then only one orientation is signaled. The cortical network selects the orientation

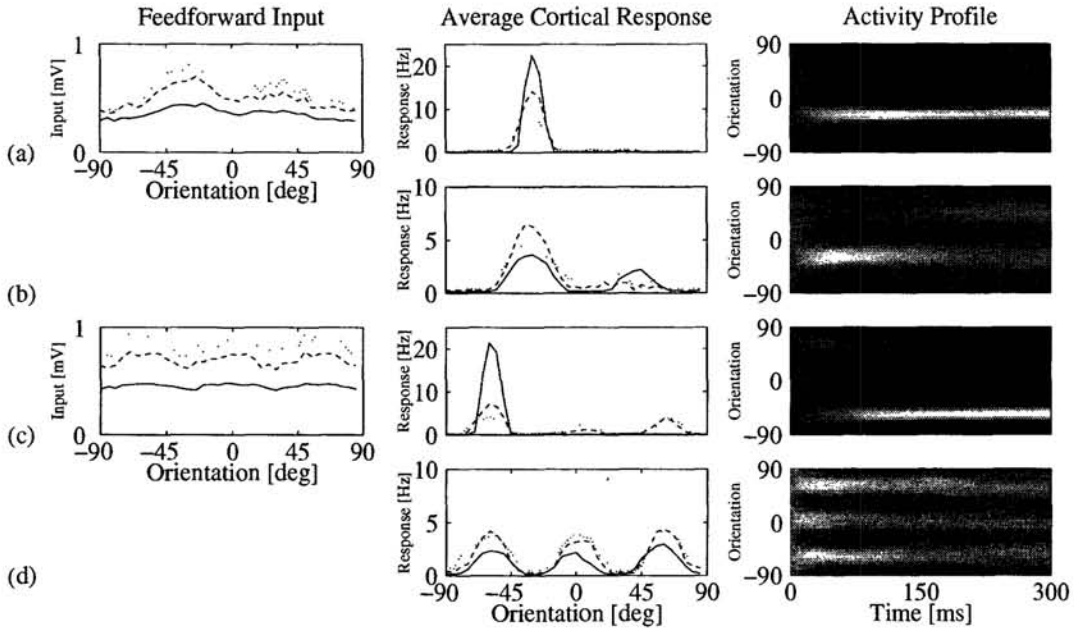

Figure 2: The response of the cortical amplifier model with static (a,c) and fast depressing recurrent synapses (b, d). In both models the feed-forward synapses are fast depressing. In the left column the feed-forward input is shown, that is same for both models. Two types of stimuli were applied. The first stimulus consists of a stronger ($\alpha = -30°$) and a weaker bar ($\alpha = +30°$) (a, b); the second stimulus consists of three equal intensity bars with orientations that are separated by $60°$ (c, d). In the middle column the cortical response is shown averaged for different time windows ([0..30] dotted; [0..80] dashed; [200..300] solid line). In the right column the cortical activity profile is plotted as a function of time. Gray values indicate the activity with bright denoting high activities.

with the highest amplitude in a winner-take-all fashion. In contrast, if synaptic depression is present at the recurrent excitatory synapses, both bars are signaled in parallel (at low release probability, Fig. 2b) or after each other (high release probability, data not shown). First, those cells fire which are tuned to the orientation of the bar with the stronger intensity, and a sharply tuned response emerges at a single orientation—the network operates in a winner-take-all regime. The synapses of these highly active cells then become strongly depressed and cortical competition decreases. As the network is shifted to a more linear operation regime, the second orientation is signaled too. Note that this phenomenon—together with the observed contrast invariant tuning—cannot be reproduced by simply decreasing the static synaptic weights in the cortical amplifier model. The recurrent synaptic efficacy changes inhomogeneously in the network depending on the activity. Only the synapses of the highly active cells depress strongly, and therefore a sharply tuned response can be evoked by a bar with weak intensity. Fast synaptic depression thus behaves as a local self-regulation that modulates competition with a certain delay. This delay, and therefore the delay of the rise of the response to the second bar depends on the effective time constant $\tau_{\text{eff}}(f(t), p) = \tau_{\text{rec}}/(1 + pf(t)\tau_{\text{rec}})$ of the synaptic depression at the recurrent connections. If the depression becomes faster due to an increase in the release probability $p$, then the delay decreases. The delay also scales with the difference between the bar intensities. The closer to each other they are, the shorter the delay will be.

In Figs. 2c, d the cortical response to three bars with equal intensities is presented. Cells tuned to the presented three orientations respond in parallel if fast synaptic depression at the recurrent excitation is present (Figs. 2d). The cortical network with strong *static* recurrent synapses again fails to signal faithfully its feed-forward input. Additive noise on the

feed-forward input introduces a slight symmetry breaking and the network with static recurrent weights responds strongly at the orientation of only one of the presented bars (Fig. 2c).

In summary, our simulations revealed that a recurrent network with fast synaptic depression is capable of obtaining robust sharpening of its feed-forward input and it also responds correctly to multiple orientations. Note that other local activity dependent adaptation mechanisms, such as slow potassium current, would have similar effects as the synaptic depression on the highly orientation specific excitatory connections. An experimentally testable prediction of our model is that the response to a flashed bar with lower contrast can be delayed by masking it with a second bar with higher contrast (Fig. 2b, right). We also suggest that long range integration from outside of the classical receptive field could emerge with a similar delay. In the initial phase of the cortical response, strong local features are amplified. In the longer, second phase, recurrent competition decreases and then weak modulatory recurrent or feed-forward input has a stronger relative effect. In the following, we investigate whether this strategy is favorable from the point of view of cortical encoding.

## 3 Dynamic coding

In the previous section we have proposed that during cortical processing a highly nonlinear phase is followed by a more linear mode if we consider a short stimulus presentation or a fixation period. The simulations demonstrated that unless the recurrent competition is modulated in time, the network fails to account for more than one feature in its input. From a strictly functional point of view the question arises, why not to use weak recurrent competition during the whole processing period. We investigate this problem in an abstract signal-encoder framework

$$\vec{y} = g(\vec{x}) + \eta \,, \tag{4}$$

where $\vec{x}$ is the input to the "cortical network", $g(\vec{x})$ is a nonlinear mapping and—for the sake of simplicity—$\eta$ is additive Gaussian noise. Naturally, in a real recurrent network output noise becomes input noise because of the feedback. Here we use the simplifying assumption that only output noise is present on the transformed input signal (input noise would lead to different predictions that should be further investigated). Output noise can be interpreted as a noisy channel that projects out from, e.g., the primary visual cortex. The nonlinear transformation $g(\vec{x})$ here is considered as a functional description of a cortical amplifier network without analyzing how actually it is "implemented". Considering orientation selectivity, the signal $\vec{x}$ can be interpreted as a vector of intensities (or contrasts) of edges with different orientations. Edges which are not present have zero intensity. The coding capacity of a realistic neural network is limited. Among several other noise sources, this limitation could arise from imprecision in spike timing and a constraint on the maximal or average firing rate.

The input-output mapping $g(\vec{x})$ of a cortical amplifier network is approximated with the soft-max function

$$g_i(\vec{x}) = \frac{\exp(\beta x_i)}{\sum_i \exp(\beta x_i)} \,. \tag{5}$$

The $\beta$ parameter can be interpreted as the level of recurrent competition. As $\beta \to 0$ the network operates in a more linear mode, while $\beta \to \infty$ puts it into a highly nonlinear winner-take-all mode. In all cases the average activity in the network is constrained which has been suggested to minimize metabolic costs [5]. Let us consider a factorizing input distribution,

$$p(\vec{x}) = \frac{1}{Z}\Pi_i \exp\left(\frac{-x_i^\alpha}{\xi}\right) \quad \text{for } x \geq 0 \,, \tag{6}$$

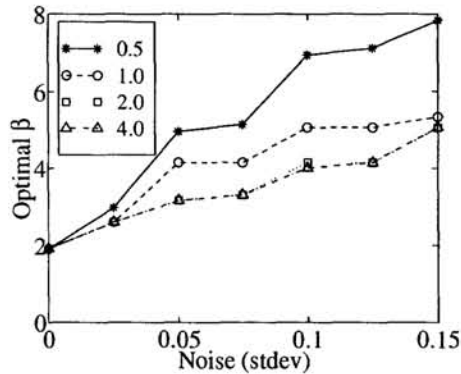

Figure 3: The optimal competition parameter $\beta$ as a function of the standard deviation of the Gaussian output noise $\eta$. The optimal $\beta$ is calculated for highly super-Gaussian, Gaussian, and sub-Gaussian stimulus densities. The sparsity parameter $\alpha$ is indicated in the legend.

where the exponent $\alpha$ determines the sparsity of the probability density function, $Z$ is a normalizing constant, and $\xi$ determines the variance. If $\alpha = 2$, the input density is the positive half of a multivariate Gaussian distribution. With $\alpha > 2$ the signal distribution becomes sub-Gaussian, and with $\alpha < 2$ it becomes super-Gaussian.

For optimal processing in *time* one needs to gain the maximal information about the signal for any increasing time window. Let us assume that the stimulus is static and it is presented for a limited time. As time goes ahead after stimulus onset, the time window for the encoding and the read-out mechanism increases. During a longer period more samples of the noisy network output are available, and thus the output noise level decreases with time. We suggest that the optimal competition parameter $\beta^{\mathrm{opt}}$—at which the mutual information between input $\vec{x}$ and output $\vec{y}$ (Eq. 4) is maximized—depends on the noise level. As the noise decreases with time, $\beta$ or the recurrent cortical competition should also change during cortical processing. To demonstrate this idea, the mutual information is calculated numerically for a three-dimensional state space.

One might expect that at higher noise levels the highest information transfer can be obtained if the typical and salient features are strongly amplified. Note that this is only true if the standard deviation of the noise scales sub-linearly with activity, which is true for an additive noise process as well as Poisson firing. As noise decreases (e.g., with increasing the time window for estimation), the level of competition should decrease distributing the available resources (e.g., spikes) among more units and letting the network respond to finer details at the input. Investigating the level of optimal competition $\beta$ as a function of the standard deviation of the output noise (Fig. 3) this intuition is indeed justified. The optimal $\beta$ scales with the standard deviation of the additive noise process. Comparing signal distributions with the same variance but with different sparsity exponents $\alpha$, we find that the sparser the signal distribution is, the higher the optimal competition becomes, because multiple features are unlikely to be present at the same time if the input distribution is sparse. By enforcing competition, the optimal encoding strategy also generates an activity distribution where only few units fire for a presented stimulus. Since edges with different orientations form a sparse distributed representation of natural scenes [8], our work suggests that a strongly competitive visual cortical network could achieve a better performance on our visual environment than a simple linear network would do.

We can now interpret our simulation results presented in the Section 2 from a functional point of view and give a prediction for the dynamics of the recurrent cortical competition. Noting that the output noise is decreasing with increasing time-window for encoding, the cortical competition should also decrease following a similar trajectory as presented in Fig. 3. If competition is low and static, then the cumulative mutual information between input and output would converge only slowly towards the overall information that is available in the stimulus. If the competition is high during the whole observation period, then after a fast rise the cumulative mutual information would saturate well below the possible

maximum. If the level of competition is dynamic, and it decreases from an initially highly competitive state, then the network obtains maximal information transfer in time.

One may argue that the *valuable* information about the signals mainly depends on the interest of the observer. Considering an encoding system for one variable it has been suggested that in a highly attentive state the recurrent competition increases [10]. In the view of our results we would refine this statement by suggesting that competition increases or decreases depending on the level of visual detail the observer pays attention to. Whenever representation of small details is also required, reducing competition is the optimal strategy given enough bandwidth.

In summary, using a detailed model for an orientation hypercolumn in V1 we have demonstrated that sharp contrast invariant tuning and faithful representation of multiple features can be achieved by a recurrent network if the recurrent competition decreases in time after stimulus onset. The model predicts that the cortical response to weak details in the stimulus emerges with a delay if a second stronger feature is also present. The modulation from, e.g., outside of the classical receptive field also has a delayed effect on cortical activity. Our study within an abstract framework revealed that weakening the recurrent cortical competition on a fast time scale is functionally advantageous, because a maximal amount of information can be transmitted in any time window after stimulus onset.

**Acknowledgments** Supported by the Boehringer Ingelheim Fonds (C. P.), by the German Science Foundation (DFG grant GK 120-2) and by Wellcome Trust 050080/Z/97.

## Footnotes

\*Current address: Epigenomics GmbH, Kastanienallee 24, D-10435 Berlin, Germany

# References

[1] L. F. Abbott, J. A. Varela, K. Sen, and S. B. Nelson. Synaptic depression and cortical gain control. *Science*, 275:220–224, 1997.

[2] P. Adorján, J.B. Levitt, J.S. Lund, and K. Obermayer. A model for the intracortical origin of orientation preference and tuning in macaque striate cortex. *Vis. Neurosci.*, 16:303–318, 1999.

[3] P. Adorján, C. Piepenbrock, and K. Obermayer. Contrast adaptation and infomax in visual cortical neurons. *Rev. Neurosci.*, 10:181–200, 1999. ftp://ftp.cs.tu-berlin.de/pub/local/ni/papers/adp99-contrast.ps.gz.

[4] Ö. B. Artun, H. Z. Shouval, and L. N. Cooper. The effect of dynamic synapses on spatiotemporal receptive fields in visual cortex. *Proc. Natl. Acad. Sci.*, 95:11999–12003, 1998.

[5] R. Baddeley. An efficient code in V1? *Nature*, 381:560–561, 1996.

[6] R. Ben-Yishai, R. Lev Bar-Or, and H. Sompolinsky. Theory of orientation tuning in visual cortex. *Proc. Natl. Acad. Sci.*, 92:3844–3848, 1995.

[7] M. Carandini and D. L. Ringach. Predictions of a recurrent model of orientation selectivity. *Vision Res.*, 37:3061–3071, 1997.

[8] D. J. Field. What is the goal of sensory coding. *Neural Comput.*, 6:559–601, 1994.

[9] D. H. Hubel and T. N. Wiesel. Receptive fields, binocular interaction and functional architecture in cat's visual cortex. *J. Physiol.*, 165:559–568, 1962.

[10] D. K. Lee, L. Itti, C. Kock, and J. Braun. Attention activates winner-take-all competition among visual filters. *Nat. Neurosci.*, 2:375–381, 1999.

[11] D. C. Somers, S. B. Nelson, and M. Sur. An emergent model of orientation selectivity in cat visual cortical simple cells. *J. Neurosci.*, 15:5448–65, 1995.

[12] H. Sompolinsky and R. Shapley. New perspectives on the mechanisms for orientation selectivity. *Curr. Op. in Neurobiol.*, 7:514–522, 1997.

[13] T. W. Troyer, A. E. Krukowski, N. J. Priebe, and K. D. Miller. Contrast-invariant orientation tuning in visual cortex: Feedforward tuning and correlation-based intracortical connectivity. *J. Neurosci.*, 18:5908–5927, 1998.

[14] M. V. Tsodyks and H. Markram. The neural code between neocortical pyramidal neurons depends on neurotransmitter release probability. *Proc. Natl. Acad. Sci.*, 94:719–723, 1997.
